# Privacy-Preserving Belief Propagation and Sampling

**Michael Kearns, Jinsong Tan, and Jennifer Wortman**
Department of Computer and Information Science
University of Pennsylvania, Philadelphia, PA 19104

## Abstract

We provide provably privacy-preserving versions of belief propagation, Gibbs sampling, and other local algorithms — distributed multiparty protocols in which each party or vertex learns only its final local value, and absolutely nothing else.

## 1 Introduction

In this paper we provide provably *privacy-preserving* versions of belief propagation, Gibbs sampling, and other local message-passing algorithms on large distributed networks. Consider a network of human social contacts, and imagine that each party would like to compute or estimate their probability of having contracted a contagious disease, which depends on the exposures to the disease of their immediate neighbors in the network. If network participants (or their proxy algorithms) engage in standard belief propagation, each party would learn their probability of exposure conditioned on any evidence — and a great deal more, including information about the exposure probabilities of their neighbors. Obviously such leakage of non-local information is highly undesirable in settings where we regard each party in the network as a self-interested agent, and privacy is paramount. Other examples include inference problems in distributed military sensor networks (where we would like the "capture" of one sensor to reveal as little non-local state information as possible), settings where networks of financial organizations would like to share limited information, and so on.

By a privacy-preserving version of inference (for example), we informally mean a protocol in which each party learns their conditional probability of exposure to the disease *and absolutely nothing else*. More precisely, anything a party can efficiently compute after having participated in the protocol, they could have efficiently computed *alone* given only the value of their conditional probability — thus, the protocol leaked no additional information beyond its desired outputs. Classical and powerful tools from cryptography [6] provide solutions to this problem, but with the significant drawback of entirely *centralizing* the privacy-preserving computation. Put another way, the straightforward solution from cryptography requires every party in the network to have the ability to broadcast to all others, and the resulting algorithm may bear little resemblance to standard belief propagation. Clearly this is infeasible in settings where the network is very large and entirely distributed, where individuals may not even know the size of the overall network, much less its structure and the identity of its constituents. While there has been work on minimizing the number of messages exchanged in *centralized* privacy-preserving protocols [9], ours are the first results that preserve the local communication structure of *distributed* algorithms like belief propagation.

Our protocols are faithful to the network topology, requiring only the passing of messages between parties separated by one or two hops in the network. Furthermore, our protocols largely preserve the algebraic structure of the original algorithms (for instance, the sum-product structure of belief propagation) and enjoy all the correctness guarantees of the originals (such as exact inference in trees for belief prop or convergence of Gibbs sampling to the joint distribution). Our technical methods show how to blend tools from cryptography (secure multiparty computation and blindable encryption) with local message-passing algorithms in a way that preserves the original computations, but in which all messages appear to be randomly distributed from the viewpoint of any individual.

All results in this paper apply to the "semi-honest" or "honest but curious" model in the cryptography literature, in which participants obediently execute the protocol but may attempt to infer non-private information from it. We expect that via the use of zero-knowledge proof techniques, our protocols may be strengthened to models in which participants who deviate from the protocol are detected.

## 2 Background and Tools from Cryptography

### 2.1 Secure Multiparty Function Computation

Let $f(x_1, \ldots, x_k)$ be any function on $k$ inputs. Imagine a scenario in which there are $k$ distinct parties, each in possession of exactly one of these inputs (that is, party $i$ initially knows $x_i$) and the $k$ parties would like to jointly compute the value of $f(x_1, \ldots, x_k)$. Perhaps the simplest protocol would have all parties share their private inputs and then individually compute the value of $f$. However, in many natural settings, we would like the parties to be able to perform this joint computation in a *privacy-preserving* fashion, with each party revealing as little as possible about their private input. Simple examples include voting — we would all like to learn the results of the election without having to broadcast our private votes — and the so-called "Millionaire's Problem" in which two individuals would like to learn who is wealthier, without revealing their precise wealth to each other. If a trusted "third party" is available, one solution would be to provide the private inputs to them, and have them perform the computation in secrecy, only announcing the final result. The purpose of the theory of secure multiparty function computation [6] is to show that under extremely general circumstances, a third party is surprisingly unnecessary.

Note that it is typically inevitable that *some* information is revealed just by the result of the computation of $f$ itself. For example, in the Millionaire's Problem, there is no avoiding the poorer party learning a lower bound on the richer's wealth (namely, the poorer party's wealth). The goal is thus to *reveal nothing beyond what it implied by the value of $f$*.

To formalize this notion in a complexity-theoretic framework, let us assume without loss of generality that each input $x_i$ is $n$ bits in length. We make the natural and common assumptions that the function $f$ can be computed in time polynomial in $kn$, and that each party's computational resources are bounded by a polynomial in $n$. We (informally) define a *protocol* $\Pi$ for the $k$ parties to compute $f$ to be a specific mechanism by which the parties exchange messages and perform computations, ending with every party learning the value $y = f(x_1, \ldots, x_k)$. One (uninteresting) protocol is the one in which each party sends their private inputs to all others, and every party computes $y$ alone.

**Definition 1** [1] *Let $\Pi$ be any protocol for the $k$ parties to jointly compute the value $y = f(x_1, \ldots, x_k)$ from their $n$-bit private inputs. We say that $\Pi$ is* privacy-preserving *if for every $1 \leq i \leq k$, anything that party $i$ can compute in time polynomial in $n$ following the execution of $\Pi$, they could also compute in polynomial time given only their private input $x_i$ and the value $y$.*

In other words, whatever information party $i$ is able to obtain from their view of the execution of protocol $\Pi$, it does not let them efficiently compute anything they couldn't efficiently compute just from being told the final output $y$ of $\Pi$ (and their private input $x_i$). This captures the notion that while $y$ itself may "leak" some information about the other private inputs $x_j$, the protocol $\Pi$ yields nothing further.[2] Further, for the following theorem we can consider more general vector outputs and randomized functionalities, which we need for our technical results.

**Theorem 1** *(See e.g. [6]) Let $f(x_1, \ldots, x_k) = (y_1, \ldots, y_k)$ be any (possibly randomized) $k$-input, $k$-output functionality that can be computed in polynomial time. Then under standard cryptographic assumptions,* [3] *there exists a polynomial time privacy-preserving protocol $\Pi$ for $f$ (that is, a protocol in which party $i$ learns nothing not already implied by their private input $x_i$ and private output $y_i$).*

This remarkable and important theorem essentially says that whatever a population can jointly compute, it can jointly compute with arbitrary restrictions on who learns what. A powerful use of vector outputs is to enforce knowledge asymmetries on the parties. For instance, in the Millionaire's Problem, by defining one player's output to always be nil, we can ensure that this player learns *absolutely nothing* from the protocol, while the other learns which player is wealthier.

The proof of Theorem 1 is *constructive*, providing an algorithm to transform any polynomial circuit into a polynomial-time privacy-preserving protocol for $k$ parties. As discussed in the introduction, this theorem can be immediately applied to (say) belief propagation to yield *centralized* privacy-preserving protocols for inference; our contribution is preserving the highly distributed, local message-passing structure of belief propagation and similar algorithms.

## 2.2 Public-Key Encryption with Blinding

The second cryptographic primitive that we shall require is standard public-key encryption with an additional property known as *blinding*. A standard public-key cryptosystem allows any party to generate a pair of *keys* $(P, S)$, which we can think of as $k$-bit strings; $k$ is often called the *security parameter*. Associated with the *public key* $P$ there is a (possibly probabilistic) *encryption function* $E_P$ and associated with the *secret or private key* $S$ there is a (deterministic) *decryption function* $D_S$. Informally, the system should have the following security properties:

- For any $n$-bit $x$, the value of the function $E_P(x)$ can be computed in polynomial time from inputs $x$ and $P$. Similarly, $D_S(y)$ can be computed efficiently given $y$ and $S$.
- For any $n$-bit input $x$, $D_S(E_P(x)) = x$. Thus, decryption is the inverse of encryption.
- For any $n$-bit $x$, it is hard for a party knowing only the public key $P$ and the encryption $E_P(x)$ to compute $x$. [4]

Thus, in such a scheme, anyone knowing the public key of Alice can efficiently compute and send encrypted messages to Alice, but only Alice, who is the sole party knowing her private key, can decrypt those messages. Such cryptosystems are widely believed to exist and numerous concrete proposals have been examined for decades. As one specific example that allows probabilistic encryption of individual bits, let the public key consist of an integer $N = p \cdot q$ that is the product of two $k/2$-bit randomly generated prime numbers $p$ and $q$, as well as a number $z$ that has the property that $z$ is not equal to $x^2 \mod N$ for any $x$. It is easy to generate such $(N, z)$ pairs. In order to encrypt a 0, one simply chooses $x$ at random and lets the encryption be $y = x^2 \mod N$, known as a *quadratic residue*. In order to encrypt a 1, one instead sends $y = x^2 \cdot z \mod N$, which is guaranteed to not be a quadratic residue. It is not difficult to show that given the factors $p$ and $q$ (which constitute the secret key), one can efficiently compute whether $y$ is a quadratic residue and thus learn the decrypted bit. Furthermore, it is widely believed that decryption is actually equivalent to factoring $N$, and thus intractable without the secret key.

This simple public-key cryptosystem also has the additional *blinding* property that we will require. Given only the public key $(N, z)$ and an encrypted bit $y$ as above, it is the case that for any value $w$, $w^2 y \mod N$ is a quadratic residue if and only if $y$ is a quadratic residue, and that furthermore $w^2 y \mod N$ is uniformly distributed among all (non-)quadratic residues if $y$ is a (non-)quadratic residue. Thus, a party knowing only Alice's public key can nevertheless take any bit encrypted for Alice and generate random re-encryptions of that bit *without* needing to actually know the decryption. We refer to this operation as *blinding* an encrypted bit.

## 3 Privacy-Preserving Belief Propagation

In this section we briefly review the standard algorithm for belief propagation on trees [10] and outline how to run this algorithm in a privacy-preserving manner such that each variable learns only its final marginals and no additional new information that is not implied by these marginals.

In standard belief propagation, we are given an undirected graphical model with vertex set $\mathcal{X}$ for which the underlying network is a tree. We denote by $\mathcal{V}(X_i)$ the set of possible values of $X_i \in \mathcal{X}$,

and by $\mathcal{N}(X_i)$ the set of $X_i$'s neighbors. For each $X_i \in \mathcal{X}$, we are given a non-negative potential function $\psi_i$ over possible values $x_i \in \mathcal{V}(X_i)$. Similarly, for each pair of adjacent vertices $X_i$ and $X_j$, we are given a non-negative potential function $\psi_{i,j}$ over joint assignments to $X_i$ and $X_j$.

The main inductive phase of the belief propagation algorithm is the message-passing phase. At each step, a node $X_i$ computes a message $\mu_{i \rightarrow j}$ to send to some $X_j \in \mathcal{N}(X_i)$. This message is indexed by all possible assignments $x_j \in \mathcal{V}(X_j)$, and is defined inductively by

$$\mu_{i \rightarrow j}(x_j) = \sum_{x_i \in \mathcal{V}(X_i)} \psi_i(x_i)\psi_{i,j}(x_i, x_j) \prod_{X_k \in \mathcal{N}(X_i)\backslash X_j} \mu_{k \rightarrow i}(x_i). \tag{1}$$

Belief propagation follows the so-called message-passing protocol, in which any vertex of degree $d$ that has received the incoming messages from any $d-1$ of its neighbors can perform the computation above in order to send an outgoing message to its remaining neighbor. Eventually, the vertex will receive a message back from this last neighbor, at which point it will be able to calculate messages to send to its remaining $d-1$ neighbors. The protocol begins at the leaves of the tree, and it follows from standard arguments that until all nodes have received incoming messages from all of their neighbors, there must be some vertex that is ready to compute and send a new message. The message-passing phase ends when all vertices have received messages from all of their neighbors.

Once vertex $X_i$ has received all of its incoming messages, the marginal distribution $\mathbf{P}$ is proportional to their product. That is, if $x_i$ is any setting to $X_i$, then

$$\mathbf{P}[X_i = x_i] \propto \psi_i(x_i) \prod_{X_j \in \mathcal{N}(X_i)} \mu_{j \rightarrow i}(x_i). \tag{2}$$

When there is *evidence* in the network, represented as a partial assignment $\vec{e}$ to some subset $E$ of the variables, we can simply hard-wire this evidence into the potential functions $\psi_j$ for each $X_j \in E$. In this case it is well-known that the algorithm computes the conditional marginals $\mathbf{P}[X_i = x_i | E = \vec{e}]$. For a more in-depth review of belief propagation, see Yedidia et al. [13] or Chapter 8 of Bishop [1].

### 3.1 Mask Propagation and the Privacy-Preserving Protocol

We assume that at the beginning of the privacy-preserving protocol, each node $X_i$ knows its own individual potential function $\psi_i$, as well as the joint potential functions $\psi_{i,j}$ for all $X_j \in \mathcal{N}(X_i)$. Recall that our fundamental privacy goal is to allow each vertex $X_i$ to compute its own marginal distribution $\mathbf{P}[X_i = x_i]$ (or $\mathbf{P}[X_i = x_i | E = \vec{e}]$ if there is evidence), but *absolutely nothing else*. In particular, $X_i$ should not be able to compute the values of any of the incoming messages from its neighbors. Knowledge of $\mu_{j \rightarrow i}(x_i)$, for example, along with $\mu_{i \rightarrow j}$ and $\psi_{i,j}$, may give $X_i$ information about the marginals over $X_j$, a clear privacy violation. We thus must somehow prevent $X_i$ from being able to "read" any of its incoming messages — nor even its own outgoing messages — yet still allow each variable to learn its own set of marginals at the end. To accomplish this we combine tools from secure multiparty function computation with a method we call "mask propagation", in which messages remain "masked" (that is, provably unreadable) to the vertices at all times. The keys required to unmask the messages are generated locally as the computation propagates through the tree, thus preserving the original communication pattern of the standard (non-private) algorithm.

Before diving into the secure protocol, we first must fix conventions regarding the encoding of numerical values. We will assume throughout that all potential function values, all message values and all the required products computed by the algorithm can be represented as $n$-bit natural numbers and thus fall in $Z_N = \{0, \dots, N-1\}$ where $N = 2^n$. As expressed by Equation (2), marginal probabilities are obtained by taking products of such $n$-bit numbers and then normalizing to obtain finite-precision real-valued numbers in the range $[0, 1]$. It will be convenient to think of values in $Z_N$ as elements of the cyclic group of order $N$ with addition and subtraction modulo $N$. In particular, we will make frequent use of the following simple fact: for any fixed $x \in Z_N$, if $r \in Z_N$ is chosen randomly among all $n$-bit numbers, then $x + r \mod N$ is also distributed randomly among all $n$-bit numbers. We can think of the random value $r$ as "masking" or hiding the value of $x$ to a party that does not know $r$, while leaving it readable to a party that does.

Let us now return to the message-passing phase of the algorithm described by Equation (1), and let us focus on the computation of $\mu_{i \rightarrow j}$ for a fixed setting $x_j$ of $X_j$. For the secure version of the algorithm, we make the following inductive message and knowledge assumptions:

- For each $X_\ell \in \mathcal{N}(X_i) \backslash X_j$, and for each setting $x_i$ of $X_i$, $X_i$ has already obtained a masked version of $\mu_{\ell \to i}(x_i)$:

$$\mu_{\ell \to i}(x_i) + \beta_{j,\ell}(x_i) \mod N \tag{3}$$

  where $\beta_{j,\ell}(x_i)$ is uniformly distributed in $Z_N$.
- $X_i$ knows only the sum in Equation (3) (which again is uniformly distributed in $Z_N$ and thus meaningless by itself), and does not know the masking values $\beta_{j,\ell}(x_i)$.
- Vertex $X_j$ knows only the masking values $\beta_{j,\ell}(x_i)$, and not the sum in Equation (3).

For all leaf nodes, these assumptions hold trivially at the start of the protocol, providing the base case for the induction. Now under these informational assumptions, vertex $X_i$ knows the set $I_i = \{\mu_{\ell \to i}(x_i) + \beta_{j,\ell}(x_i) \mod N : X_\ell \in \mathcal{N}(X_i) \backslash X_j, \ x_i \in \mathcal{V}(X_i)\}$ while vertex $X_j$ knows the set $I_j = \{\beta_{j,\ell}(x_i) \mod N : X_\ell \in \mathcal{N}(X_i) \backslash X_j, \ x_i \in \mathcal{V}(X_i)\}$.

Let us first consider the case in which $X_j$ is not a leaf node and thus has neighbors other than $X_i$ itself. In order to complete the inductive step, it will be necessary for each $X_k \in \mathcal{N}(X_j) \backslash X_i$ to provide a set of masking values $\beta_{k,i}(x_j)$ so that $X_j$ can obtain a set of masked messages of the form $\mu_{i \to j}(x_j) + \beta_{k,i}(x_j)$. Here we focus on a single neighbor $X_k$ of $X_j$.

Vertex $X_k$ privately generates a masking value $\beta_{k,i}(x_j)$ that is uniformly distributed in $Z_n$. It is clear that, ignoring privacy concerns, $X_i$ and $X_j$ together could compute $\psi_i(x_i)\psi_{i,j}(x_i, x_j)\prod_{X_\ell \in \mathcal{N}(X_i)\backslash X_j} \mu_{\ell \to i}(x_i)$ for each fixed pair $x_i$ and $x_j$. Thus from their joint inputs $I_i$, $I_j$, and $\beta_{k,i}(x_j)$, ignoring privacy, $X_i$, $X_j$, and $X_k$ could compute:

$$\left( \sum_{x_i \in \mathcal{V}(X_i)} \psi_i(x_i)\psi_{i,j}(x_i, x_j) \prod_{X_\ell \in \mathcal{N}(X_i)\backslash X_j} \mu_{\ell \to i}(x_i) \right) + \beta_{k,i}(x_j) \mod N$$
$$= \mu_{i \to j}(x_j) + \beta_{k,i}(x_j) \mod N \tag{4}$$

Since this expression can be computed jointly by $X_i$, $X_j$ and $X_k$ without privacy considerations, Theorem 1 establishes that we can construct an efficient protocol for them to compute it securely, allowing $X_j$ to learn *only the value of the expression* in Equation (4), while $X_i$ and $X_k$ learn no new information at all (i.e. nil output). Note that this expression, due to the presence of the unknown masking value $\beta_{k,i}(x_j)$, is a uniform randomly distributed number in $Z_n$ from $X_j$'s point of view.

After this masking process has been completed for all $X_k \in \mathcal{N}(X_j) \backslash X_i$, we will have begun to satisfy the inductive informational assumptions a step further in the propagation: for each neighbor $X_k$ of $X_j$ excluding $X_i$, $X_j$ will know a masked version of $\mu_{i \to j}(x_j)$ in which the masking value $\beta_{k,i}(x_j)$ is known only to $X_k$. $X_j$ will obtain masked messages in a similar manner from all but one of its other neighbors in turn, and for all of its other values, until the inductive assumptions are fully satisfied at $X_j$. Every value received by $X_i$, $X_j$, and $X_k$ during the above protocol is distributed uniformly at random in $Z_n$ from the perspective of its recipient, and thus conveys no information.

It remains to consider the case in which $X_j$ is a leaf node. In this case, there is no need to satisfy the inductive assumptions at the next level, as the propagation ends at the leaves. Furthermore, it is acceptable for $X_j$ to learn its incoming messages directly, since these messages will be implied by its final marginal. From their joint input $I_i$ and $I_j$, it is clear that $X_i$ and $X_j$ together could compute $\mu_{i \to j}(x_j)$ as given in Equation (1). Thus by Theorem 1, we can construct a protocol for them to efficiently compute this value in such a way that $X_j$ learns only $\mu_{i \to j}(x_j)$ and $X_i$ learns nothing.

At the end of the message-passing phase, each internal (non-leaf) node $X_i$ will know a set of masked messages from each of its neighbors. In particular, for each pair $X_j, X_\ell \in \mathcal{N}(X_i)$, for each $x_i \in \mathcal{V}(X_i)$, $X_i$ will know the values of $\mu_{j \to i}(x_i) + \beta_{\ell,j}(x_i)$. Ignoring privacy concerns, it is the case that $X_i$ and any pair of its neighbors could compute the marginal of $X_i$ in Equation (2). Invoking Theorem 1 again, we can construct an efficient protocol for $X_i$ and this pair of neighbors to together compute the marginals such that $X_i$ learns only the marginals and the neighbors learn nothing.

Each leaf vertex $X_i$ will be in possession of its unmasked messages $\mu_{j \to i}(x_i)$ for every $x_i \in \mathcal{V}(X_i)$ from its neighbor $X_j$, and can easily compute its marginals as given in Equation (2) without having learned anything not already implied by its initial potential functions and the marginals themselves.

We use **PrivateBeliefProp**$(T)$ to denote the algorithm above when applied to a particular tree $T$. The full proof of the following is omitted, but follows the logic sketched in the preceding sections.

**Theorem 2** *Under standard cryptographic assumptions,* **PrivateBeliefProp***(T) allows every variable $X_i$ to compute its own marginal distribution $\mathbf{P}[X_i]$ and nothing else (that is, nothing not already computable in polynomial time from only $\mathbf{P}[X_i]$ and the initial potential functions). Direct communication occurs only between variables who are immediate neighbors or two steps away in T, and secure function computation is never invoked on sets of more than three variables.* [5]

We briefly note a number of extensions to Theorem 2 and the methods described above.

**Loopy Belief Propagation:** Theorem 2 can be extended to privacy-preserving loopy belief propagation on graphs that contain cycles. Because of the protocol's faithfulness to the original algorithm, the same convergence and correctness claims hold as in standard loopy belief propagation [7].

**Computing Only Partial Information:** Allowing a variable to learn its exact numerical marginal distribution may actually convey a great deal of information. We might instead only want each variable to learn, for instance, whether its probability of taking on a given value is greater than 0.1 or not. Theorem 2 can easily be generalized to allow each variable to learn only any partial information about its own marginal.

**Privacy-Preserving Junction Tree:** The protocol can also be modified to perform privacy-preserving belief propagation on a junction tree [11]. Here it is necessary to take *intra-clique* privacy into account in order to enforce that variables can learn only their own marginals and not, for example, the marginals of other nodes within the same clique.

**NashProp and Other Message-Passing Algorithms:** The methods described here can also be applied to provide privacy-preserving versions of the NashProp algorithm [8], allowing players in a multiparty game to jointly compute and draw actions from a Nash equilibrium, with each player learning only his own action and nothing else.[6] We are investigating more general applications of our methods to a broad class of message-passing algorithms that would include many others.

## 4    Privacy-Preserving Gibbs Sampling

We now move on to the problem of secure Gibbs sampling on an undirected graphical model $G$. The local potential functions accompanying $G$ can be preprocessed to obtain conditional distributions for each variable given a setting of all its neighbors (Markov blanket). Thus we henceforth assume that each variable has access to its local conditional distribution, which it will be convenient to represent in a particular tabular form. To simplify presentation, we will assume each variable is binary, taking on values in $\{0, 1\}$, but this assumption is easy to relax.

If a node $X_i$ is of degree $d$, the conditional distribution of $X_i$ given a particular assignment to its neighbors will be represented by a table $T_i$ with $2^d$ rows and $d + 1$ columns. The first $d$ columns range over all $2^d$ possible assignments $\vec{x}$ to $\mathcal{N}(X_i)$, while the final column contains the numerical value $\mathbf{P}[X_i = 1 | \mathcal{N}(X_i) = \vec{x}]$. We will use $T_i(\vec{x})$ to denote the value $\mathbf{P}[X_i = 1 | \mathcal{N}(X_i) = \vec{x}]$ stored in the $d + 1$st column in the row corresponding to the assignment $\vec{x}$.

With this notation, the standard (non-private) Gibbs sampling algorithm [4, 2] can be easily described. After choosing an initial assignment to all of the variables in $G$ (for instance, uniformly at random), the algorithm repeatedly resamples values for individual variables conditioned on the current values of their neighbors. More precisely, at each step, a variable $X_i$ is chosen for resampling. Its current value is replaced by randomly drawing value 1 with probability $T_i(\vec{x})$ and value 0 with probability $1 - T_i(\vec{x})$ where $\vec{x}$ is the current set of assignments to $\mathcal{N}(X_i)$.

To implement a privacy-preserving variant of Gibbs sampling, we must solve the following cryptographic problem: how can a set of vertices communicate with their neighbors in order to repeatedly resample their values from their conditional distributions given their neighbors' current assignments, without learning any information except their own final values at the end of the process and anything that is implied by these values? Again, we would like to accomplish this with limited communication so that no vertex is required to communicate with a vertex more than two hops away.

In order for each variable to learn only its *final* sampled value after some number of iterations, and not its intermediate resampled values (which may be enough to provide a good approximation of the marginal distribution on the variable), we first provide a way of distributing the current value of a vertex so that it cannot be learned by any vertex in isolation. One way of accomplishing this is by assigning each vertex $X_i$ a "distinguished neighbor" $N^*(X_i)$. $X_i$ will hold one bit $b_i$ while $N^*(X_i)$ will hold a second bit $b_i'$ such that the current value of $X_i$ is $b_i \oplus b_i'$.

Using such an encoding, there is a simple but relatively inefficient construction for privacy-preserving Gibbs sampling that uses only secure multiparty function computation, but that invokes Theorem 1 on entire neighborhoods of the graph. In graphs with high degree, this requires broadcast communication between a large number of parties, which we would like to avoid. Here we describe a much more communication-efficient protocol using blinded encryption. For concreteness the reader may imagine below that we are using the blindable cryptosystem based on quadratic residues described in Section 2.2, though other choices are possible.

We begin by describing a sub-protocol for preprocessing the table $T_i$ before resampling begins. Let $S$ be the $2^d$ indices of the rows of the table $T_i$. For ease of notation, we will refer to the $d$ neighbors of $X_i$ as $V_1, \ldots, V_d$. The purpose of the sub-protocol is for $X_i$ and its neighbors to compute a random permutation $\pi$ of $S$ (which can be thought of as a random permutation of the rows of $T_i$) in such a way that during the protocol, each $V_j \in \mathcal{N}(X_i)$ learns only the sets $\{\pi(\vec{x}) : V_j = 0\}$ and $\{\pi(\vec{x}) : V_j = 1\}$ and $X_i$ learns nothing.

The sub-protocol is quite simple. First each neighbor $V_j$ of $X_i$ encrypts column $j$ of $T_i$ using its own public key and passes the encrypted column to $X_i$. Next $X_i$ encrypts column $d + 1$ using its own public key. $X_i$ then concatenates the $d + 1$ encrypted columns together to form an encrypted version of $T_i$ in which column $j$ is encrypted using the public key of $V_j$ for $1 \le j \le d$ and column $d + 1$ is encrypted using the public key of $X_i$. $X_i$ then takes the resulting table, randomly permutes the rows, and blinds (randomly re-encrypts) each entry using the appropriate public keys (i.e. the key of $V_j$ for column $j$ where $1 \le j \le d$ and its own public key for column $d + 1$). At this point, $X_i$ sends the resulting table to its distinguished neighbor $N^*(X_i)$.

The purpose of the blinding steps here is to prevent parties from tracking correspondences between cleartext and encrypted table entries. For instance, without blinding above, $N^*(X_i)$ could reconstruct the permutation chosen by $X_i$ by seeing how its own encrypted values have been rearranged. Now from the perspective of $N^*(X_i)$, $d$ columns of the table will look like uniformly distributed random bits. $N^*(X_i)$ will still be able to decrypt the column of the table that corresponds to its own values, but it will become clear that decrypting this column alone cannot yield useful information.

In the next step in the protocol, $N^*(X_i)$ re-encrypts column $d + 1$ of the table with its own public key. It then randomly permutes the rows of the table, blinds each entry using the appropriate public keys (those of $V_j$ for columns $1 \le j \le d$ and its own for column $d + 1$), and sends the updated table back to $X_i$. At this point, every entry in the table will look random bits to $X_i$. Each column $j$ will be encrypted by the public key of $V_j$, with the exception of the final column, which will be encrypted by both $X_i$ and $N^*(X_i)$. Call this new table $T_i'$.

Once these encrypted tables have been computed for each node, we begin the main Gibbs sampling protocol. We inductively assume that at the start of each step, for each $X_j \in \mathcal{X}$, the current value of $X_j$ is distributed between $X_j$ and $N^*(X_j)$. At the end of the step, the only information that has been learned is the new value of a particular node $X_i$, but distributed between $X_i$ and $N^*(X_i)$.

Consider a neighbor $V_j$ of $X_i$. $V_j$ can decrypt column $j$ of $T_i'$ in order to learn which rows correspond to its value being 0 and which rows correspond to its values being 1. While $V_j$ alone does not know what its current value is, $V_j$ and $N^*(V_j)$ could compute it together, and thus could together figure out which rows of the permutation correspond to $V_j$'s current value. By Theorem 1, since there is a way for them to compute this information ignoring privacy, we can construct an efficient protocol for $V_j$, $N^*(V_j)$, and $X_i$ to perform this computation such that $X_i$ learns only the rows that correspond to $V_j$'s value (and in particular does not learn what this value is), while $V_j$ and $N^*(V_j)$ learn nothing.

After this secure computation of partitions has been completed for all neighbors of $X_i$, $X_i$ will be able to compute the intersection of the subsets of rows it has received from each neighbor. This intersection will be a single row corresponding to the current values of all nodes in $\mathcal{N}(X_i)$. Initially, $X_i$ will not be able to decrypt any of the entries in this row. However, $X_i$ and $N^*(X_i)$ could together

decrypt the value in column $d + 1$, use this value in order to sample $X_i$'s new value according to the appropriate distribution, and distribute the new value between themselves. Calling upon Theorem 1 once again, this means that we can construct an efficient protocol for $X_i$ and $N^*(X_i)$ to together complete these computations in such a way that they only learn the new bits $b_i$ and $b_i'$ respectively.

Each time the value of a node $X_i$ is resampled, $X_i$ and $N^*(X_i)$ repeat the process of blinding and permuting the rows of $T_i'$. This prevents $X_i$ and its neighbors from learning how frequently they take on different values throughout the sampling process. After the value of each node has been privately resampled sufficiently many times, we can use one final application of secure multi-party computation between each node $X_i$ and its distinguished neighbor $N^*(X_i)$ to allow $X_i$ to learn its final value.

As with standard Gibbs sampling, we also need to specify a schedule by which vertices in the Markov network will have their values updated, as well as the number of iterations of this schedule, which will in turn determine how close the sampled distribution is to the true joint (stationary) distribution. Since our interests are in privacy considerations only, let us use **PrivateGibbs** to refer to the protocol described above when applied to any fixed Markov network, combined with some fixed updating schedule (such as random or a fixed ordering) and some number $r$ of iterations.

**Theorem 3** *Under standard cryptographic assumptions*[7], **PrivateGibbs** *computes a sample from the joint distribution after $r$ iterations, with every variable learning its own value and nothing else. Direct communication occurs only between variables who are immediate neighbors or two steps away, and secure function computation is never invoked on sets of more than three variables.*

The full proof is again omitted, but largely follows the sketch above. We note that **PrivateGibbs** enjoys an even stronger privacy property — even if any subset of parties collude by combining their post-protocol views, they can learn nothing not implied by their combined sampled values. Furthermore, any convergence guarantees that hold for standard Gibbs sampling [4, 5] with the same updating schedule will also hold for the secure version.

## Footnotes

[1] We state this definition informally, as the complete technical definition is somewhat lengthy and adds little intuition. It involves both formalizing the notion of a multiparty computation protocol, as well as defining the "view" of an individual party of a specific execution of the protocol. The definition involves computational indistinguishability of probability distributions since the protocols may often use randomization.

[2] Our definition of privacy does not imply that *coalitions* of parties cannot together compute additional information. In the extended version of this paper, we discuss the difficulty of achieving this stronger notion of privacy with any protocol that uses a truly distributed method of computation.

[3] An example would be the existence of trapdoor permutations [6].

[4]This is often formalized by asserting that the distribution of the encryption is computationally indistinguishable from true randomness in time polynomial in $n$ and $k$.

[5]Since the application of standard secure function computation requires broadcast among all participants, it is a feature of the algorithm that it limits such invocations to three parties at a time.

[6]See work by Dodis et al. [3] and Teague [12] for more on privacy-preserving computation in game theory.

[7]An example would be intractability of recognizing quadratic residues.

# References

[1] C. Bishop. *Pattern Recognition and Machine Learning*. Springer, 2006.

[2] G. Casella and E. George. Explaining the Gibbs sampler. *The American Statistician*, 46:167–174, 1992.

[3] Y. Dodis, S. Halevi, and T. Rabin. A cryptographic solution to a game theoretic problem. In *CRYPTO*, pages 112–130, 2000.

[4] S. Geman and D. Geman. Stochastic relaxation, Gibbs distributions, and the Bayesian restoration of images. *IEEE Transactions on Pattern Analysis and Machine Intelligence*, 6:721–741, 1984.

[5] A. Gibbs. Bounding convergence time of the Gibbs sampler in Bayesian image restoration. *Biometrika*, 87:749–766, 2000.

[6] O. Goldreich. *Foundations of Cryptography, Volume 2*. Cambridge University Press, 2004.

[7] A. Ihler, J. Fisher III, and A. Willsky. Loopy belief propagation: Convergence and effects of message errors. *Journal of Machine Learning Research*, 6:905–936, 2005.

[8] M. Kearns, M. Littman, and S. Singh. Graphical models for game theory. In *Uncertainty in Artificial Intelligence*, 2001.

[9] M. Naor and K. Nissim. Communication preserving protocols for secure function evaluation. In *ACM Symposium on Theory of Computing*, pages 590–599, 2001.

[10] J. Pearl. *Probabilistic Reasoning in Intelligent Systems: Networks of Plausible Inference*. Morgan Kaufmann, 1988.

[11] P. Shenoy and G. Shafer. Axioms for probability and belief-function propagation. In *Uncertainty in Artificial Intelligence*, pages 169–198, 1990.

[12] V. Teague. Selecting correlated random actions. In *Financial Cryptography*, pages 181–195, 2004.

[13] J. Yedidia, W. Freeman, and Y. Weiss. Understanding belief propagation and its generalizations. In *Exploring Artificial Intelligence in the New Millennium*. Morgan Kaufmann, 2003.

